# Joint Analysis of Time-Evolving Binary Matrices and Associated Documents

[1]**Eric Wang**, [1]**Dehong Liu**, [1]**Jorge Silva**, [2]**David Dunson** and [1]**Lawrence Carin**
[1]Electrical and Computer Engineering Department, Duke University
[2]Statistics Department, Duke University
`{eric.wang,dehong.liu,jg.silva,lawrence.carin}@duke.edu`
`dunson@stat.duke.edu`

## Abstract

We consider problems for which one has incomplete binary matrices that evolve with time (*e.g.*, the votes of legislators on particular legislation, with each year characterized by a different such matrix). An objective of such analysis is to infer structure and inter-relationships underlying the matrices, here defined by latent features associated with each axis of the matrix. In addition, it is assumed that documents are available for the entities associated with at least one of the matrix axes. By jointly analyzing the matrices and documents, one may be used to inform the other within the analysis, and the model offers the opportunity to predict matrix values (*e.g.*, votes) based only on an associated document (*e.g.*, legislation). The research presented here merges two areas of machine-learning that have previously been investigated separately: incomplete-matrix analysis and topic modeling. The analysis is performed from a Bayesian perspective, with efficient inference constituted via Gibbs sampling. The framework is demonstrated by considering all voting data and available documents (legislation) during the 220-year lifetime of the United States Senate and House of Representatives.

## 1 Introduction

There has been significant recent research on the analysis of incomplete matrices [10, 15, 1, 12, 13, 18]. Most analyses have been performed under the assumption that the matrix is real. There are interesting problems for which the matrices may be binary; for example, reflecting the presence/absence of links on nodes of a graph, or for analysis of data associated with a series of binary questions. One may connect an underlying real matrix to binary (or, more generally, integer) observations via a probit or logistic link function; for example, such analysis has been performed in the context of analyzing legislative roll-call data [6]. A problem that has received less attention concerns the analysis of time-evolving matrices. The specific motivation of this paper involves binary questions in a legislative setting; we are interested in analyzing such data over many legislative sessions, and since the legislators change over time, it is undesirable to treat the entire set of votes as a single matrix. Each piece of legislation (question) is unique, but it is desirable to infer inter-relationships and commonalities over time. Similar latent groupings and relationships exist for the legislators. This general setting is also of interest for analysis of more-general social networks [8].

A distinct line of research has focused on analysis of documents, with topic modeling constituting a popular framework [4, 2, 17, 3, 11]. Although the analysis of matrices and documents has heretofore been performed independently, there are many problems for which documents and matrices may be coupled. For example, in addition to a matrix of links between websites or email sender/recipient data, one also has access to the associated documents (website and email content). By analyzing the matrices and documents simultaneously, one may infer inter-relationships about each. For example, in a factor-based model of matrices [8], the associated documents may be used to relate matrix factors to topics/words, providing insight from the documents about the matrix, and *vice versa*.

To the authors' knowledge, this paper represents the first joint analysis of time-evolving matrices and associated documents. The analysis is performed using nonparametric Bayesian tools; for example, the truncated Dirichlet process [7] is used to jointly cluster latent topics and matrix features. The framework is demonstrated through analysis of large-scale data sets. Specifically, we consider binary vote matrices from the United States Senate and House of Representatives, from the first congress in 1789 to the present. Documents of the legislation are available for the most recent 20 years, and those are also analyzed jointly with the matrix data. The quantitative predictive performance of this framework is demonstrated, as is the power of this setting for making qualitative assessments of large-scale and complex joint matrix-document data.

## 2 Modeling Framework

### 2.1 Time-evolving binary matrices

Assume we are given a set of binary matrices, $\{\mathbf{B}_t\}_{t=1,\tau}$, with $\mathbf{B}_t \in \{0,1\}^{N_y^{(t)} \times N_x^{(t)}}$. The number of rows and columns, respectively $N_y^{(t)}$ and $N_x^{(t)}$, may vary with time. For example, for the legislative roll-call data consider below, time index $t$ corresponds to year and the number of pieces of legislation and legislators changes with time ($e.g.$, for the historical data considered for the United States congress, the number of states and hence legislators changes as the country has grown).

Using a modeling framework analogous to that in [6], the binary matrix has a probit-model generative process, with $\mathbf{B}_t(i,j) = 1$ if $\mathbf{X}_t(i,j) > 0$, and $\mathbf{B}_t(i,j) = 0$ otherwise, and the latent real matrix is defined as

$$\mathbf{X}_t(i,j) = <\boldsymbol{y}_i^{(t)}, \boldsymbol{x}_j^{(t)}> +\beta_i^{(t)} + \alpha_j^{(t)} + \epsilon_{i,j}^{(t)} \tag{1}$$

where $< \cdot, \cdot >$ denotes a vector inner product, and $\epsilon_{i,j}^{(t)} \sim \mathcal{N}(0,1)$. The random effects are drawn $\beta_i^{(t)} \sim \mathcal{N}(0, \lambda_\beta^{-1})$ and $\alpha_j^{(t)} \sim \mathcal{N}(0, \lambda_\alpha^{-1})$, with $\lambda_\alpha \sim \mu_\alpha \delta_\infty + (1-\mu_\alpha)\text{Gamma}(a,b)$ and $\lambda_\beta \sim \mu_\beta \delta_\infty + (1-\mu_\beta)\text{Gamma}(a,b)$; $\delta_\infty$ is a point measure at infinity, corresponding to there not being an associated random effect. The probability of whether there is a random effect is controlled by $\mu_\beta$ and $\mu_\alpha$, each of which is drawn from a beta distribution.

Random effect $\alpha_j$ is motivated by our example application, for which the index $j$ denotes a specific piece of legislation that is voted upon; this parameter reflects the "difficulty" of the vote, and if $|\alpha_j|$ is large, then all people are likely to vote one way or the other (an "easy" vote), while if $\alpha_j^{(t)}$ is small the details of the legislator (defined by $\boldsymbol{y}_i^{(t)}$) and legislation (defined by $\boldsymbol{x}_j^{(t)}$) strongly impact the vote. In previous political science Bayesian analysis [6] researchers have simply set $\mu_\beta = 1$ and $\mu_\alpha = 0$, but here we consider the model in a more-general setting, and infer these relationships.

Additionally, in previous Bayesian analysis [6] the dimensionality of $\boldsymbol{y}_i^{(t)}$ and $\boldsymbol{x}_j^{(t)}$ has been set (usually to one or two). In related probabilistic matrix factorization (PMF) applied to real matrices [15, 12], priors/regularizers are employed to constrain the dimensionality of the latent features. Here we employ the sparse binary vector $\boldsymbol{b} \in \{0,1\}^K$, with $b_k \sim \text{Bernoulli}(\pi_k)$, and $\pi_k \sim \text{Beta}(c/K, d(K-1)/K)$, for $K$ set to a large integer. By setting $c$ and $d$ appropriately, this favors that most of the components of $\boldsymbol{b}$ are zero (imposes sparseness). Specifically, by integrating out the $\{\pi_k\}_{k=1,K}$, one may readily show that the number of non-zero components in $\boldsymbol{b}$ is a random variable drawn from $\text{Binomial}(K, c/(c + d(K-1)))$, and the expected number of ones in $\boldsymbol{b}$ is $cK/[c + d(K-1)]$. This is related to a draw from a truncated beta-Bernoulli process [16].

We consider two types of matrix axes. Specifically, we assume that each row corresponds to a person/entity that may be present for matrix $t+1$ and matrix $t$. It is assumed here that each column corresponds to a question (in the examples, a piece of legislation), and each question is unique. Since the columns are each unique, we assume $\boldsymbol{x}_j^{(t)} = \boldsymbol{b} \circ \hat{\boldsymbol{x}}_j^{(t)}$, $\hat{\boldsymbol{x}}_j^{(t)} \sim \mathcal{N}(\mathbf{0}, \gamma_x^{-1}\mathbf{I}_K)$, $\gamma_x \sim \text{Gamma}(e,f)$, where $\circ$ denotes the pointwise/Hadamard vector product. If the person/entity associated with the $i$th row at time $t$ is introduced for the first time, its associated feature vector is similarly drawn $\boldsymbol{y}_i^{(t)} = \boldsymbol{b} \circ \hat{\boldsymbol{y}}_i^{(t)}$, $\hat{\boldsymbol{y}}_i^{(t)} \sim \mathcal{N}(\mathbf{0}, \gamma_y^{-1}\mathbf{I}_K)$, with $\gamma_y \sim \text{Gamma}(e,f)$. However, assuming $\boldsymbol{y}_i^{(t)}$ is already drawn (person/entity $i$ is active prior to time $t+1$), then a simple auto-regressive model is used to draw $\boldsymbol{y}_i^{(t+1)}$: $\boldsymbol{y}_i^{(t+1)} = \boldsymbol{b} \circ \hat{\boldsymbol{y}}_i^{(t+1)}$, $\hat{\boldsymbol{y}}_i^{(t+1)} \sim \mathcal{N}(\hat{\boldsymbol{y}}_i^{(t)}, \xi^{-1}\mathbf{I}_K)$, with $\xi \sim \text{Gamma}(g,h)$. The prior on $\xi$ is set to favor small/smooth changes in the features of an individual on consecutive years.

This model constitutes a relatively direct extension of existing techniques for real matrices [15, 12]. Specifically, we have introduced a probit link function and a simple auto-regression construction to

impose statistical correlation in the traits of a person/entity at consecutive times. The introduction of the random effects $\alpha_j$ and $\beta_i$ has also not been considered within much of the machine-learning matrix-analysis literature, but the use of $\alpha_j$ is standard in political science Bayesian models [6]. The principal modeling contribution of this paper concerns how one may integrate such a time-evolving binary-matrix model with associated documents.

## 2.2 Topic model

The manner in which the topic modeling is performed is a generalization of latent Dirichlet allocation (LDA) [4]. Assume that the documents of interest have words drawn from a vocabulary $\mathbf{V} = \{w_1, \ldots, w_V\}$. The $k$th topic is characterized by a distribution $\boldsymbol{p}_k$ on words ("bag-of-words" assumption), where $\boldsymbol{p}_k \sim \mathrm{Dir}(\alpha_V/V, \ldots, \alpha_V/V)$. The generative model draws $\{\boldsymbol{p}_k\}_{k=1,T}$ once for each of the $T$ possible topics.

Each document is characterized by a probability distribution on topics, where the $\boldsymbol{c}_l \sim \mathrm{Dir}(\alpha_T/T, \ldots, \alpha_T/T)$ corresponds to the distribution across $T$ topics for document $l$. The generative process for drawing words for document $l$ is to first (and once) draw $\boldsymbol{c}_l$ for document $l$. For word $i$ in document $l$, we draw a topic $z_{il} \sim \mathrm{Mult}(\boldsymbol{c}_l)$, and then the specific word is drawn from a multinomial with probability vector $\boldsymbol{p}_{z_{il}}$.

The above procedure is like the standard LDA [4], with the difference manifested in how we handle the Dirichlet distributions $\mathrm{Dir}(\alpha_V/V, \ldots, \alpha_V/V)$ and $\mathrm{Dir}(\alpha_T/T, \ldots, \alpha_T/T)$. The Dirichlet distribution draws are constituted via Sethuraman's construction [14]; this allows us to place gamma priors on $\alpha_V$ and $\alpha_T$, while retaining conjugacy, permitting analytic Gibbs' sampling (we therefore get a full posterior distribution for all model parameters, while most LDA implementations employ a point estimate for the document-dependent probabilities of topics). Specifically, the following hierarchical construction is used for draws from $\mathrm{Dir}(\alpha_V/V, \ldots, \alpha_V/V)$ (and similarly for $\mathrm{Dir}(\alpha_T/T, \ldots, \alpha_T/T)$):

$$\boldsymbol{p}_k = \sum_{h=1}^{\infty} a_h \delta_{\theta_h} \,, \quad a_h = U_h \prod_{n<h}(1 - U_n) \,, \quad U_h \sim \mathrm{Beta}(1, \alpha_V) \,, \quad \theta_h \sim \sum_{w=1}^{V} \frac{1}{V} \delta_w \qquad (2)$$

The probability mass $a_h$ is associated with component $\theta_h \in \{1, \ldots, V\}$ of the probability vector. The infinite sum is truncated, analogous to the truncated stick-breaking representation of the Dirichlet process [9].

## 2.3 Joint analysis of matrices and documents

Section 2.1 discusses how we model time-evolving binary matrices, and Section 2.2 describes our procedure for implementing topic models. We now put these two models together. Specifically, we consider the case for which there is a document $\mathcal{D}_j^{(t)}$ of words associated with the $j$th column at time $t$; in our example below, this will correspond to the $j$th piece of legislation in year $t$. It is possible that we may have documents associated with the matrix rows as well (*e.g.*, speeches for the $i$th legislature), but in our model development (and in our examples), documents are only assumed present for the columns.

For column $j$ at time $t$, we have both a feature vector $\boldsymbol{x}_j^{(t)}$ (for the matrix) and a distribution on topics $\boldsymbol{c}_j^{(t)}$ (for the document $\mathcal{D}_j^{(t)}$), and these are now coupled; the remainder of the matrix and topic models are unchanged. We define a set of atoms $\{\boldsymbol{c}_m^*, \boldsymbol{\mu}_m^*, \zeta_m^*\}_{m=1,M}$. The atoms $\boldsymbol{\mu}_m^*$ are drawn from $\mathcal{N}(\mathbf{0}, \gamma_x^{-1} \mathbf{I}_K)$, again with a gamma prior placed on $\gamma_x$, and $\zeta_m^*$ are also drawn from a gamma distribution; the $\boldsymbol{c}_m^*$ are drawn iid from $\mathrm{Dir}(\alpha_T/T, \ldots, \alpha_T/T)$, using the Dirichlet distribution construction as above. To couple the pair $(\boldsymbol{x}_j^{(t)}, \boldsymbol{c}_j^{(t)})$, we draw indicator variable $u_{jt}$ as

$$u_{jt} \sim \sum_{m=1}^{M} b_m \delta_m \,, \quad b_m = C_m \prod_{i<m}(1 - C_i) \,, \quad C_m \sim \mathrm{Beta}(1, \eta) \qquad (3)$$

with a gamma prior again placed on $\eta$ (with $C_M = 1$). The pair $(\boldsymbol{x}_j^{(t)}, \boldsymbol{c}_j^{(t)})$ is now defined by $\boldsymbol{x}_j^{(t)} = \boldsymbol{b} \circ \hat{\boldsymbol{x}}_j^{(t)}$, with $\hat{\boldsymbol{x}}_j^{(t)} \sim \mathcal{N}(\boldsymbol{\mu}_{u_{jt}}^*, \zeta_{u_{jt}}^{*}{}^{-1} \mathbf{I}_K)$. Further, $\boldsymbol{c}_j^{(t)}$ is set to $\boldsymbol{c}_{u_{jt}}^*$.

This construction clusters the columns, with the clustering mechanism defined by a truncated stick-breaking representation of the Dirichlet process [9]. The components $\{\boldsymbol{\mu}_m^*, \zeta_m^*\}_{m=1,M}$ define a

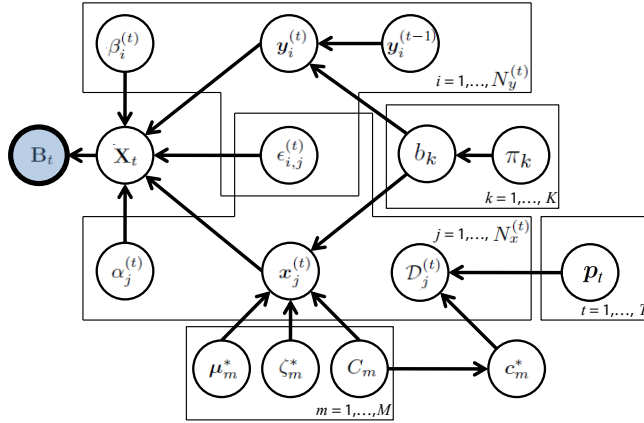

Figure 1: Graphical representation of the model, with the hyperparameters omitted for simplicity. The plates indicate replication, and the filled circle around $\mathbf{B}_t$ indicates it is observed.

Gaussian mixture model (GMM) in matrix-column feature space, while the $\{c_m^*\}_{m=1,M}$ define a set of $M$ probability vectors over topics, with one such vector associated with each of the aforementioned GMM mixture components. The truncated Dirichlet process infers how many mixture components are needed to represent the data.

In this construction, each of the matrix columns is associated with a distribution on topics (based upon which mixture component it is drawn from). This provides powerful interpretative insights between the latent features in the matrix model and the words from the associated documents. Further, since the topic and matrix models are constituted jointly, the topics themselves are defined as to be best matched to the characteristics of the matrix (*vis-a-vis* simply modeling the documents in isolation, which may yield topics that are not necessarily well connected to what matters for the matrices). A graphical representation of the model is shown in Figure 1.

There are several extensions one may consider in future work. For example, for simplicity the GMM in column feature space is assumed time-independent. One may consider having a separate GMM for each time (year) $t$. Further, we have not explicitly imposed time-dependence in the topic model itself, and this may also be considered [2, 11]. For the examples presented below on real data, despite these simplifications, the model seems to perform well.

## 2.4 Computations

The posterior distribution of all model parameters has been computed using Gibbs sampling; the detailed update equations are provided as supplemental material at `http://sites.google.com/site/matrixtopics/`. The first 1000 Gibbs iterations were discarded as burn-in followed by 500 collection iterations.The truncation levels on the model are $T = 20$, $M = 10$, $K = 30$, and the number of words in the vocabulary is $V = 5249$. Hyperparameters were set as $a = b = e = f = 10^{-6}$, $c = d = 1$, $g = 10^3$, and $h = 10^{-3}$. None of these parameters have been optimized, and "reasonable" related settings yield very similar results.

We have performed joint matrix and text analysis considering the United States Congress voting records (and, when available, the document associated with the legislation); we consider both the House of Representatives (House) and Senate, from 1789-2008. Legislation documents and meta-data (bill sponsorship, party affiliation of voters, etc.) are available for sessions 101–110 (1989-2008). For the legislation, stop words were removed using a common stopword list (the 514 stop words are posted at `http://sites.google.com/site/matrixtopics/`, and the corpus was stemmed using a Porter stemmer). These data are available from `www.govtrack.us` and from the Library of Congress `thomas.loc.gov` (votes, text and metadata), while the votes dating from 1789 are at `voteview.com`. A binary matrix is manifested by mapping all "affirmative" vote codes (*e.g.*, "Yea", "Yes", "Present") to one, and "negative" codes (*e.g.*, "Nay","No","Not Present") to zero. Not all legislatures are present to vote on a given piece of legislation, and therefore missing data are manifested naturally. It varies from year to year, but typically 4% of the votes are missing in a given year.

We implemented our proposed model in non-optimized Matlab. Computations were performed on a PC with a 3.6GHz CPU and 4GB memory. A total of 11.5 hours of CPU time are required for

analysis of Senate sessions 101-110 (1989-2008), and 34.6 hours for House sessions 101-110; in both cases, this corresponds to joint analysis of both votes and text (legislation). If we only analyze the votes, 15.5 hours of CPU are required for Senate session 1-110 (1789-2008), and 62.1 hours for House 1-110 respectively (the number of legislators in the House is over four times larger than that for the Senate).

## 3 Experiments

### 3.1 Joint analysis of documents and votes

We first consider the joint analysis of the legislation (documents) and votes in the Senate, for 1989-2008. A key aspect of this analysis is the clustering of the legislation, with legislation $j$ at time $t$ mapped to a cluster (mixture component), with each mixture component characterized by a distribution across latent topics $c_j^{(t)}$, and a latent feature $x_j^{(t)}$ for the associated matrix analysis (recall Section 2.3). Five dominant clusters were inferred for these data. Since we are running a Gibbs sampler, and the cluster index changes in general between consecutive iterations (because the index is exchangeable), below we illustrate the nature of the clusters based upon the last Gibbs iteration.

The dimensionality of the features was inferred to be $\|b\|_0 = 5$ (on average, across the Gibbs collection), but two dimensions dominated for the legislation feature vectors $x_j^{(t)}$. In Figure 2 we present the inferred distributions of the five principal mixture components (clusters). The cluster index and the indices of the features are arbitrary; we, for example, number the clusters from 1 to 5 for illustrative simplicity.

In Figure 2 we depict the distribution of topics $c_m^*$ associated with each of the five clusters, and in Figure 3 we list the ten most probable words associated with each of the topics. By examining the topic characteristics in Figure 3, and the cluster-dependent distribution of topics, we may assign words/topics to the latent features $x_j^{(t)}$ that are linked to the associated matrix, and hence to the vote itself. For example, clusters 1 and 4, which are the most separated in latent space (top row in Figure 2), share a very similar support over topics (bottom row in Figure 2). These clusters appear to be associated with highly partisan topics, specifically taxes (topics 11 and 15) and health/Medicare/Social Security (topics 12 and 16), as can be seen by considering the topic-dependent words in Figure 3. Based upon the voting data and the party of the legislation sponsor (bill author), cluster 1 (red) appears to represent a Republican viewpoint on these topics, while cluster 4 (blue) appears to represent a Democratic viewpoint. This distinction will play an important role in predicting the votes on legislation based on the documents, as discussed below in Section 3.2.

In Figure 4 (last plot) we present the estimated density functions for the random-effect parameters $\beta_i^{(t)}$ and $\alpha_j^{(t)}$ (estimated from the Gibbs collection iterations). Note that $p(\beta)$ is much more tightly concentrated around zero than $p(\alpha)$. In the political science literature [6] (in which the legislation/documents have not been considered), researchers simply just set $\beta = 0$, and therefore only assume random effects on the legislation, but not on the senators/congressman. Our analysis appears to confirm that this simplification is reasonable.

### 3.2 Matrix prediction based on documents

There has been significant recent interest in the analysis of matrices, particularly in predicting matrix entries that are missing at random [10, 15, 1, 12, 13, 18]. In such collaborative-filtering research, the views of a subset of individuals on a movie, for example, help inform predictions on ratings of people who have not seen the movie (but a fraction of the people must have seen every movie). However, in the problem considered here, these previous models are not applicable: prediction of votes on a new legislation $\mathcal{L}_N$ requires one to relate $\mathcal{L}_N$ to votes on previous legislation $\mathcal{L}_1, \ldots, \mathcal{L}_{N-1}$, but in the absence of *any* prior votes on $\mathcal{L}_N$; this corresponds to estimating an entire column of the vote matrix). The joint analysis of text (legislation) and votes, however, offers the ability to relate $\mathcal{L}_N$ to $\mathcal{L}_1, \ldots, \mathcal{L}_{N-1}$, by making connections via the underlying topics of the legislation (documents), even in the absence of any votes for $\mathcal{L}_N$.

To examine this predictive potential, we performed joint analysis on all votes and legislation (documents) in the US Senate from 1989-2007. Through this process, we yielded a model very similar to that summarized in Figures 2-4. Using this model, we predict votes on new legislation in 2008, based on the documents of the associated legislation (but using no vote information on this new legislation). To do this, the *mixture* of topics learned from 1989-2007 data are assumed fixed (each topic characterized by a distribution over words), and these fixed topics are used in the analysis of

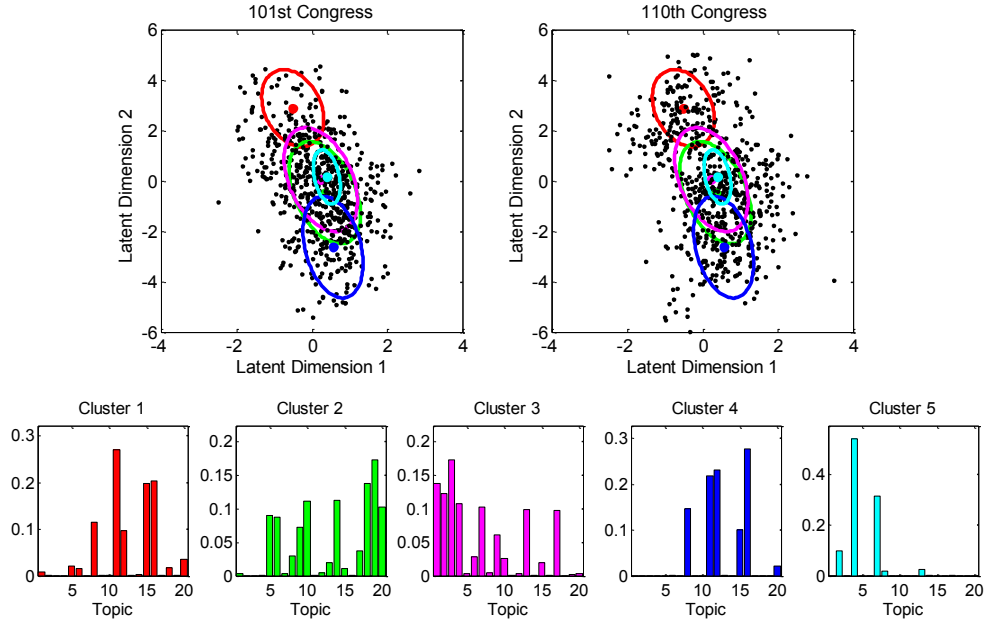

Figure 2: Characteristics of the five principal mixture components (clusters) associated with Senate data, based upon joint analysis of the documents and the associated vote matrix. **Top row:** Principal two dimensions of the latent matrix features $\boldsymbol{x}_j^{(t)}$, with the ellipses denoting the standard deviation about the mean of the five clusters. The points reflect specific legislation, with results shown for the 101st and 110th Congresses. The colors of the ellipses are linked to the colors of the topic distributions. **Bottom row:** Distribution of topics $\boldsymbol{c}_m^*$ for the five clusters (number indices arbitrary). $T = 20$ topics are considered, and each cluster is characterized by a distribution over topics $\boldsymbol{c}_m^*$ (bottom row), as well as an associated feature (top row) for the matrix.

| Topic 1 | Topic 2 | Topic 3 | Topic 4 | Topic 5 | Topic 6 | Topic 7 | Topic 8 | Topic 9 | Topic 10 |
|---|---|---|---|---|---|---|---|---|---|
| annual | military | fuel | military | public | law | defense | penalty | employee | foreign |
| research | defense | transport | defense | research | violate | civilian | expense | public | law |
| economy | this | public | navy | transport | import | iraq | health | cost | terrorist |
| doe | product | research | air | annual | goal | train | drug | defense | criminal |
| food | expense | agriculture | guard | children | bureau | health | property | domestic | agriculture |
| sale | restore | export | research | train | commerce | cost | credit | work | justice |
| motor | public | electrical | closure | expense | registration | foreign | public | inspect | terror |
| crop | annual | forest | naval | law | reform | environment | work | bureau | engage |
| county | universal | foreign | ndaa | student | risk | air | medical | tax | economy |
| employee | independence | water | bonus | organization | list | depend | organization | build | crime |

| Topic 11 | Topic 12 | Topic 13 | Topic 14 | Topic 15 | Topic 16 | Topic 17 | Topic 18 | Topic 19 | Topic 20 |
|---|---|---|---|---|---|---|---|---|---|
| tax | tax | military | violence | tax | medicare | loan | annual | immigration | alien |
| budget | health | transportation | victim | health | tax | environment | health | juvenile | civil |
| annual | drug | safety | drug | annual | ssa | train | this | firearm | parent |
| debtor | medicaid | air | alien | cost | annual | property | public | sentence | ha |
| bankruptcy | candidate | defense | employee | law | deduct | science | defend | alien | immigrant |
| foreign | cost | health | visa | this | hospital | annual | esea | crime | labor |
| taxpayer | children | guard | youth | mail | parent | law | product | dh | criminal |
| credit | aggregate | annual | penalty | financial | bankruptcy | transportation | fcc | train | free |
| property | law | foreign | criminal | liability | debtor | high | carrier | convict | term |
| product | medical | waste | minor | loan | male | five | columbia | prison | petition |

Figure 3: Top-ten most probable words associated with the Senate-legislation topics, 1989-2008.

the documents from new legislation. In this manner, each of the new documents is mapped to one of the mixture-dependent distributions on topics $\{\boldsymbol{c}_m^*\}_{m=1,M}$. If a particular piece of legislation is mapped to cluster $m$ (with mapping based upon the words alone), it is then assumed that the latent *matrix* feature associated with the legislation is the associated cluster mean $\boldsymbol{\mu}_m^*$ (learned via the modeling of 1989-2007 data).

Once this mapping of legislation to matrix latent space is achieved, and using the senator's latent feature vector $\boldsymbol{y}_i^{(t)}$ from 2007, we may readily compute $< \boldsymbol{y}_i^{(t)}, \boldsymbol{\mu}_m^* >$, and via the probit link function the probability of a "yes" vote is quantified, for Senator $i$ on new legislation $\mathcal{L}_N$. This is the model in (1), with $\beta_i^{(t)} = 0$ and $\alpha_j^{(t)} = 0$. Based upon Figure 4 (last plot), the approximation $\beta_i^{(t)} = 0$ is reasonable. The legislation-dependent random effect $\alpha_j^{(t)}$ is expected to be important

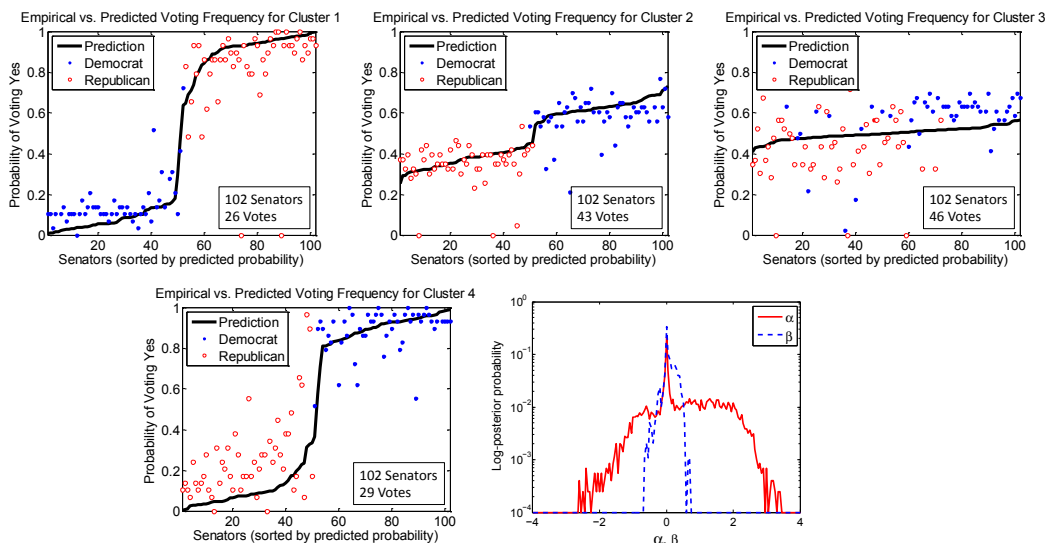

Figure 4: **First four plots:** Predicted probability of voting "Yes" given only the legislation text for 2008, based upon the model learned using vote-legislation data from 1898–2007. The dots (colored by party affiliation) show the empirical voting frequencies for all legislation in the cluster, from 2008 (not used in model). Only four clusters are utilized during session 2008, out of five inferred by the model for the overall period 1989–2007. **Last plot:** Estimated $\log p(\alpha)$ and $\log p(\beta)$. Note how $p(\beta)$ is much more sharply peaked near zero.

for legislation for which most senators vote "yes" (large positive $\alpha_j^{(t)}$) or "no" (large negative $\alpha_j^{(t)}$).

When testing the predictive quality of the model for the held-out year 2008, we assume $\alpha_j^{(t)} = 0$ (since this parameter cannot be inferred without modeling the text and votes jointly, while for 2008 we are only modeling the documents); we therefore only test the model on legislation from 2008 for which less than 90% of the senators agreed, such legislation assumed corresponding to small $|\alpha_j^{(t)}|$ (it is assumed that in practice it would be simple to determine whether a piece of legislation is likely to be near-unanimous "yes" or "no", and therefore model-based prediction of votes for such legislation is deemed less interesting).

In Figure 4 we compare the predicted, probit-based probability of a given senator voting "yes" for legislation within clusters 1-4 (see Figure 2); the points in Figure 4 represent the empirical data for each senator, and the curve represents the predictions of the probit link function. These results are deemed to be remarkably good. In Figure 4, the senators along each horizontal axis are ordered according to the probability of voting "yes".

One interesting issue that arises in this prediction concerns clusters 1 and 4 in Figure 2, and the associated predictions for the held-out year 2008, in Figure 4. Since the distributions of these clusters over topics is very similar, the documents alone cannot distinguish between clusters 1 and 4. However, we also have the sponsor of each piece of legislation, and based upon the data from 1989-2007, if a piece of legislation from 2008 is mapped to either cluster 1 or 4, it is disambiguated based upon the party affiliation of the sponsor (cluster 1 is a Republican viewpoint on these topics, while cluster 4 is a Democratic viewpoint, based upon voting records from 1989-2007).

### 3.3 Time evolution of congressman and legislation

The above joint analysis of text and votes was restricted to 1989-2008, since the documents (legislation) were only available for those years. However, the dataset contains votes on all legislation from 1789 to the present, and we now analyze the vote data from 1789-1988. Figure 5 shows snapshots in time of the latent space for voters and legislation, for the House of Representatives (similar results have been computed for the Senate, and are omitted for brevity; as supplemental material, at `http://sites.google.com/site/matrixtopics/` we present movies of how legislation and congressman evolve across all times, for both the House and Senate). Five features were inferred, with the two highest-variance features chosen for the axes. The blue symbols denote Democratic legislators, or legislation sponsored by a Democrat, and the red points correspond to Republicans. Results like these are of interest to political scientists, and allow examination of the degree of partisanship over time, for example.

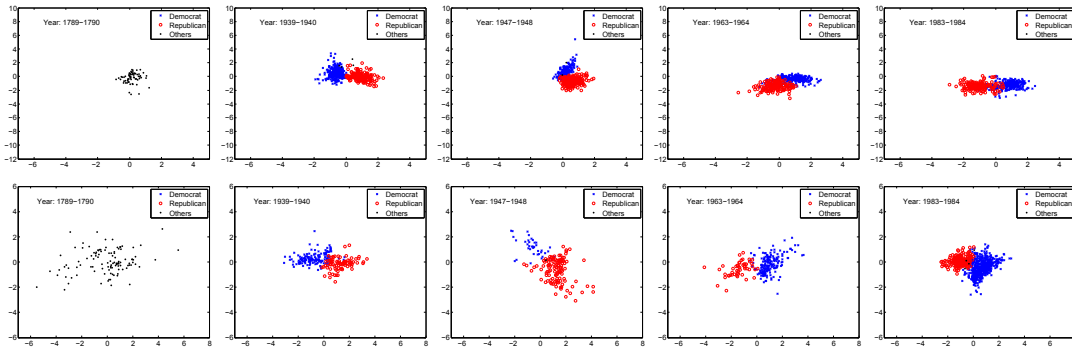

Figure 5: Congressman (top) and legislation (bottom) in latent space for sessions 1–98 of the House of Representatives. The Democrat/Republican separation is usually sharper than for the Senate, and frequently only the partisan information seems to matter. Note the gradual rotation of the red/blue blue axis. Best viewed electronically, zoomed-in.

## 3.4   Additional quantitative tests

One may ask how well this model addresses the more-classical problem of estimating the values of matrix data that are missing uniformly at random, in the absence of documents. To examine this question, we considered binary Senate vote data from 1989-2008, and removed a fraction of the votes uniformly at random, and then use the proposed time-evolving matrix model to process the observed data, and to compute the probability of a "yes" vote on all missing data (via the probit link function). If the probability is larger than 0.5 the vote is set to "yes", and otherwise it is set to "no". We compare our time-evolving model to [12], with the addition of a probit link function; for the latter we processed all 20 years as one large matrix, rather than analyzing time-evolving structure. Up to 40% missingness, the proposed model and a modified version of that in [12] performed almost identically, with an average probability of error (on the binary vote) of approximately 0.1. For greater than 40% missingness, the proposed time-evolving model manifested a "phase transition", and the probability of error increased smoothly up to 0.3, as the fraction of missing data rose to 80%; in contrast, the generalized model in [12] (with probit link) continued to yield a probability of error of about 0.1. The phase transition of the proposed model is likely manifested because the entire matrix is partitioned by year, with a linkage between years manifested via the Markov process between legislators (we don't analyze all data by one contiguous, large matrix). The phase transition is expected based on the theory in [5], when the fraction of missing data gets large enough (since the size of the contiguous matrices analyzed by the time-evolving model is much smaller than that of the entire matrix, such a phase transition is expected with less missingness than via analysis of the entire matrix at once).

While the above results are of interest and deemed encouraging, such uniformly random missingness on matrix data alone is not the motivation of the proposed model. Rather, traditional matrix-analysis methods [10, 15, 1, 12, 13, 18] are incapable of predicting votes on new legislation based on the words alone (as in Figure 4), and such models do not allow analysis of the time-evolving properties of elements of the matrix, as in Figure 5.

## 4   Conclusions

A new model has been developed for the joint analysis of time-evolving matrices and associated documents. To the authors' knowledge, this paper represents the first integration of research heretofore performed separately on topic models and on matrix analysis/completion. The model has been implemented efficiently via Gibbs sampling. A unique set of results are presented using data from the US Senate and House of Representatives, demonstrating the ability to predict the votes on new legislation, based only on the associated documents. The legislation data was considered principally because it was readily available and interesting in its own right; however, the proposed framework is of interest for many other problems. For example, the model is applicable to analysis of time-evolving relationships between multiple entities, augmented by the presence of documents (*e.g.*, links between websites, and the associated document content).

## Acknowledgement

The research reported here was supported by the US Army Research Office, under grant W911NF-08-1-0182, and the Office of Naval Research under grant N00014-09-1-0212.

# References

[1] J. Abernethy, F. Bach, T. Evgeniou, and J.-P. Vert. A new approach to collaborative filtering: operator estimation with spectral regularization. *J. Machine Learning Research*, 2009.

[2] D. M. Blei and J. D. Lafferty. Dynamic topic models. *Proceedings of the 23rd International Conference on Machine Learning*, pages 113–120, 2006.

[3] D. M. Blei and J. D. Lafferty. A correlated topic model of science. *The Annals of Applied Statistics*, 1(1):17–35, 2007.

[4] D. M. Blei, A. Y. Ng, and M. I. Jordan. Latent Dirichlet allocation. *Journal of Machine Learning Research*, 3:993–1022, 2003.

[5] E.J. Candès and T. Tao. The power of convex relaxation: Near-optimal matrix completion. *IEEE Transactions on Information Theory*, 56(5):2053–2080, 2010.

[6] J. Cinton, S. Jackman, and D. Rivers. The statistical analysis of roll call data. *Am. Political Sc. Review*, 2004.

[7] T. S. Ferguson. A bayesian analysis of some nonparametric problems. *The Annals of Statistics*, 1(2):209–230, 1973.

[8] P. D. Hoff. Multiplicative latent factor models for description and prediction of social networks. *Computational and Mathematical Organization Theory*, 2009.

[9] J. Ishwaran and L. James. Gibbs sampling methods for stick-breaking priors. *Journal of the American Statistical Association*, 96:161174, 2001.

[10] E. Meeds, Z. Ghahramani, R. Neal, and S. Roweis. Modeling dyadic data with binary latent factors. In *Advances in NIPS*, pages 977–984, 2007.

[11] I. Pruteanu-Malinici, L. Ren, J. Paisley, E. Wang, and L. Carin. Hierarchical bayesian modeling of topics in time-stamped documents. *IEEE Trans. Pattern Analysis Mach. Intell.*, 2010.

[12] R. Salakhutdinov and A. Mnih. Bayesian probabilistic matrix factorization with mcmc. In *Advances in NIPS*, 2008.

[13] R. Salakhutdinov and A. Mnih. Probabilistic matrix factorization. In *Advances in NIPS*, 2008.

[14] J. Sethuraman. A constructive definition of dirichlet priors. *Statistica Sinica*, 4:639–650, 1994.

[15] N. Srebro, J.D.M. Rennie, and T.S. Jaakkola. Maximum-margin matrix factorization. In *Advances in NIPS*, 2005.

[16] R. Thibaux and M.I. Jordan. Hierarchical beta processes and the indian buffet process. In *International Conference on Artificial Intelligence and Statistics*, 2007.

[17] H. M. Wallach. Topic modeling: beyond bag of words. *Proceedings of the 23rd International Conference on Machine Learning*, 2006.

[18] K. Yu, J. Lafferty, S. Zhu, and Y. Gong. Large-scale collaborative prediction using a nonparametric random effects model. In *Proc. Int. Conf. Machine Learning*, 2009.

